# Sequentially fitting "inclusive" trees for inference in noisy-OR networks

**Brendan J. Frey[1], Relu Patrascu[1], Tommi S. Jaakkola[2], Jodi Moran[1]**

[1] Intelligent Algorithms Lab
University of Toronto
www.cs.toronto.edu/~frey

[2] Computer Science and
Electrical Engineering
Massachusetts Institute of Technology

## Abstract

An important class of problems can be cast as inference in noisy-OR Bayesian networks, where the binary state of each variable is a logical OR of noisy versions of the states of the variable's parents. For example, in medical diagnosis, the presence of a symptom can be expressed as a noisy-OR of the diseases that may cause the symptom – on some occasions, a disease may fail to activate the symptom. Inference in richly-connected noisy-OR networks is intractable, but approximate methods (*e.g.*, variational techniques) are showing increasing promise as practical solutions. One problem with most approximations is that they tend to concentrate on a relatively small number of modes in the true posterior, ignoring other plausible configurations of the hidden variables. We introduce a new sequential variational method for bipartite noisy-OR networks, that favors *including* all modes of the true posterior and models the posterior distribution as a tree. We compare this method with other approximations using an ensemble of networks with network statistics that are comparable to the QMR-DT medical diagnostic network.

## 1 Inclusive variational approximations

Approximate algorithms for probabilistic inference are gaining in popularity and are now even being incorporated into VLSI hardware (T. Richardson, personal communication). Approximate methods include variational techniques (Ghahramani and Jordan 1997; Saul *et al.* 1996; Frey and Hinton 1999; Jordan *et al.* 1999), local probability propagation (Gallager 1963; Pearl 1988; Frey 1998; MacKay 1999a; Freeman and Weiss 2001) and Markov chain Monte Carlo (Neal 1993; MacKay 1999b). Many algorithms have been proposed in each of these classes.

One problem that most of the above algorithms suffer from is a tendency to concentrate on a relatively small number of modes of the *target distribution* (the distribution being approximated). In the case of medical diagnosis, different modes correspond to different explanations of the symptoms. Markov chain Monte Carlo methods are usually guaranteed to eventually sample from all the modes, but this may take an extremely long time, even when tempered transitions (Neal 1996) are

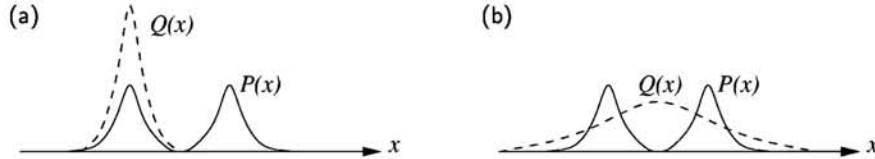

Figure 1: We approximate $P(x)$ by adjusting the mean and variance of a Gaussian, $Q(x)$. (a) The result of minimizing $D(Q||P) = \sum_x Q(x)log(Q(x)/P(x))$, as is done for most variational methods. (b) The result of minimizing $D(P||Q) = \sum_x P(x)log(P(x)/Q(x))$.

used. Preliminary results on local probability propagation in richly connected networks show that it is sometimes able to oscillate between plausible modes (Murphy *et al.* 1999; Frey 2000), but other results also show that it sometimes diverges or oscillates between implausible configurations (McEliece *et al.* 1996).

Most variational techniques minimize a cost function that favors finding the single, most massive mode, excluding less probable modes of the target distribution (*e.g.*, Saul *et al.* 1996; Ghahramani and Jordan 1997; Jaakkola and Jordan 1999; Frey and Hinton 1999; Attias 1999). More sophisticated variational techniques capture multiple modes using substructures (Saul and Jordan 1996) or by leaving part of the original network intact and approximating the remainder (Jaakkola and Jordan 1999). However, although these methods increase the number of modes that are captured, they still exclude modes.

Variational techniques approximate a target distribution $P(x)$ using a simpler, parameterized distribution $Q(x)$ (or a parameterized bound). For example, $P(disease_1, disease_2, \dots, disease_N | symptoms)$ may be approximated by a factorized distribution, $Q_1(disease_1)Q_2(disease_2) \cdots Q_N(disease_N)$. For the current set of observed symptoms, the parameters of the $Q$-distributions are adjusted to make $Q$ as close as possible to $P$.

A common approach to variational inference is to minimize a relative entropy,

$$D(Q||P) = \sum_x Q(x) \log \frac{Q(x)}{P(x)}. \tag{1}$$

Notice that $D(Q||P) \neq D(P||Q)$. Often $D(Q||P)$ can be minimized with respect to the parameters of $Q$ using iterative optimization or even exact optimization.

To see how minimizing $D(Q||P)$ may exclude modes of the target distribution, suppose $Q$ is a Gaussian and $P$ is bimodal with a region of vanishing density between the two modes, as shown in Fig. 1. If we minimize $D(Q||P)$ with respect to the mean and variance of $Q$, it will cover only one of the two modes, as illustrated in Fig. 1a. (We assume the symmetry is broken.) This is because $D(Q||P)$ will tend to infinity if $Q$ is nonzero in the region where $P$ has vanishing density.

In contrast, if we minimize $D(P||Q) = \sum_x P(x)log(P(x)/Q(x))$ with respect to the mean and variance of $Q$, it will cover *all* modes, since $D(P||Q)$ will tend to infinity if $Q$ vanishes in any region where $P$ is nonzero. See Fig. 1b.

For many problems, including medical diagnosis, it is easy to argue that it is more important that our approximation include all modes than exclude nonplausible configurations at the cost of excluding other modes. The former leads to a low number of false negatives, whereas the latter may lead to a large number of false negatives (concluding a disease is not present when it is).

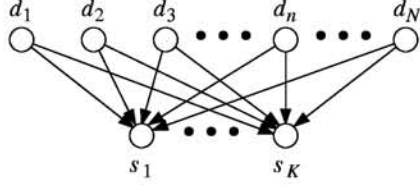

Figure 2: Bipartite Bayesian network. $s_k$s are observed, $d_n$s are hidden.

## 2 Bipartite noisy-OR networks

Fig. 2 shows a bipartite noisy-OR Bayesian network with $N$ binary hidden variables $\mathbf{d} = (d_1, \dots, d_N)$ and $K$ binary observed variables $\mathbf{s} = (s_1, \dots, s_K)$. Later, we present results on medical diagnosis, where $d_n = 1$ indicates a disease is active, $d_n = 0$ indicates a disease is inactive, $s_k = 1$ indicates a symptom is active and $s_k = 0$ indicates a symptom is inactive.

The joint distribution is

$$P(\mathbf{d}, \mathbf{s}) = \Big[\prod_{k=1}^{K} P(s_k|\mathbf{d})\Big]\Big[\prod_{n=1}^{N} P(d_n)\Big]. \tag{2}$$

In the case of medical diagnosis, this form assumes the diseases are independent.[1] Although some diseases probably do depend on other diseases, this form is considered to be a worthwhile representation of the problem (Shwe $et$ $al.$, 1991).

The likelihood for $s_k$ takes the noisy-OR form (Pearl 1988). The probability that symptom $s_k$ fails to be activated ($s_k = 0$) is the product of the probabilities that each active disease fails to activate $s_k$:

$$P(s_k = 0|\mathbf{d}) = p_{k0} \prod_{n=1}^{N} p_{kn}^{d_n}. \tag{3}$$

$p_{kn}$ is the probability that an active $d_n$ fails to activate $s_k$. $p_{k0}$ accounts for a "leak probability". $1 - p_{k0}$ is the probability that symptom $s_k$ is active when none of the diseases are active.

Exact inference computes the distribution over $\mathbf{d}$ given a subset of observed values in $\mathbf{s}$. However, if $s_k$ is not observed, the corresponding likelihood (node plus edges) may be deleted to give a new network that describes the marginal distribution over $\mathbf{d}$ and the remaining variables in $\mathbf{s}$. So, we assume that we are considering a subnetwork where all the variables in $\mathbf{s}$ are observed.

We reorder the variables in $\mathbf{s}$ so that the first $J$ variables are active ($s_k = 1$, $1 \leq k \leq J$) and the remaining variables are inactive ($s_k = 0$, $J + 1 \leq k \leq K$). The posterior distribution can then be written

$$P(\mathbf{d}|\mathbf{s}) \propto P(\mathbf{d}, \mathbf{s}) = \Big[\prod_{k=1}^{J}\Big(1 - p_{k0}\prod_{n=1}^{N} p_{kn}^{d_n}\Big)\Big]\Big[\prod_{k=J+1}^{K}\Big(p_{k0}\prod_{n=1}^{N} p_{kn}^{d_n}\Big)\Big]\Big[\prod_{n=1}^{N} P(d_n)\Big]. \tag{4}$$

Taken together, the two terms in brackets on the right take a simple, product form over the variables in $\mathbf{d}$. So, the first step in inference is to "absorb" the inactive

variables in $\mathbf{s}$ by modifying the priors $P(d_n)$ as follows:

$$P'(d_n) = \alpha_n P(d_n) \Big( \prod_{k=J+1}^{K} p_{kn} \Big)^{d_n},$$  (5)

where $\alpha_n$ is a constant that normalizes $P'(d_n)$.

Assuming the inactive symptoms have been absorbed, we have

$$P(\mathbf{d}|\mathbf{s}) \propto \Big[ \prod_{k=1}^{J} \Big( 1 - p_{k0} \prod_{n=1}^{N} p_{kn}^{d_n} \Big) \Big] \Big[ \prod_{n=1}^{N} P'(d_n) \Big].$$  (6)

The term in brackets on the left does not have a product form. The entire expression can be multiplied out to give a sum of $2^J$ product forms, and exact "QuickScore" inference can be performed by combining the results of exact inference in each of the $2^J$ product forms (Heckerman 1989). However, this exponential time complexity makes large problems, such as QMR-DT, intractable.

## 3   Sequential inference using inclusive variational trees

As described above, many variational methods minimize $D(Q||P)$, and find approximations that exclude some modes of the posterior distribution. We present a method that minimizes $D(P||Q)$ sequentially – by absorbing one observation at a time – so as to *not* exclude modes of the posterior. Also, we approximate the posterior distribution with a tree. (Directed and undirected trees are equivalent – we use a directed representation, where each variable has at most one parent.)

The algorithm absorbs one active symptom at a time, producing a new tree by searching for the tree that is closest – in the $D(P||Q)$ sense – to the product of the previous tree and the likelihood for the next symptom. This search can be performed efficiently in $\mathcal{O}(N^2)$ time using probability propagation in two versions of the previous tree to compute weights for edges of a new tree, and then applying a minimum-weight spanning-tree algorithm.

Let $T_k(\mathbf{d})$ be the tree approximation obtained after absorbing the $k$th symptom, $s_k = 1$. Initially, we take $T_0(\mathbf{d})$ to be a tree that decouples the variables and has marginals equal to the marginals obtained by absorbing the inactive symptoms, as described above.

Interpreting the tree $T_{k-1}(\mathbf{d})$ from the previous step as the current "prior" over the diseases, we use the likelihood $P(s_k = 1|\mathbf{d})$ for the next symptom to obtain a new estimate of the posterior:

$$\tilde{P}_k(\mathbf{d}|s_1,\ldots,s_k) \propto T_{k-1}(\mathbf{d}) P(s_k = 1|\mathbf{d}) = T_{k-1}(\mathbf{d}) \Big( 1 - p_{k0} \prod_{n=1}^{N} p_{kn}^{d_n} \Big)$$

$$= T_{k-1}(\mathbf{d}) - T'_{k-1}(\mathbf{d}),$$  (7)

where $T'_{k-1}(\mathbf{d}) = T_{k-1}(\mathbf{d}) \Big( p_{k0} \prod_{n=1}^{N} p_{kn}^{d_n} \Big)$ is a modified tree.

Let the new tree be $T_k(\mathbf{d}) = \prod_n T_k(d_n|d_{\pi_k(n)})$, where $\pi_k(n)$ is the index of the parent of $d_n$ in the new tree. The parent function $\pi_k(n)$ and the conditional probability tables of $T_k(\mathbf{d})$ are found by minimizing

$$D(\tilde{P}_k||T_k) = \sum_{\mathbf{d}} \tilde{P}_k(\mathbf{d}|s_1,\ldots,s_k) \log \frac{\tilde{P}_k(\mathbf{d}|s_1,\ldots,s_k)}{T_k(\mathbf{d})}.$$  (8)

Ignoring constants, we have

$$
\begin{aligned}
D(\tilde{P}_k \| T_k) &= - \sum_{\mathbf{d}} \tilde{P}_k(\mathbf{d}|s_1, \ldots, s_k) \log T_k(\mathbf{d}) \\
&= - \sum_{\mathbf{d}} \left( T_{k-1}(\mathbf{d}) - T'_{k-1}(\mathbf{d}) \right) \log \left( \prod_n T_k(d_n | d_{\pi_k(n)}) \right) \\
&= - \sum_n \left( \sum_{\mathbf{d}} \left( T_{k-1}(\mathbf{d}) - T'_{k-1}(\mathbf{d}) \right) \log T_k(d_n | d_{\pi_k(n)}) \right) \\
&= - \sum_n \left( \sum_{d_n} \sum_{d_{\pi_k(n)}} \left( T_{k-1}(d_n, d_{\pi_k(n)}) - T'_{k-1}(d_n, d_{\pi_k(n)}) \right) \log T_k(d_n | d_{\pi_k(n)}) \right).
\end{aligned}
$$

For a given structure (parent function $\pi_k(n)$), the optimal conditional probability tables are

$$
T_k(d_n | d_{\pi_k(n)}) = \beta_n \left( T_{k-1}(d_n, d_{\pi_k(n)}) - T'_{k-1}(d_n, d_{\pi_k(n)}) \right), \tag{9}
$$

where $\beta_n$ is a constant that ensures $\sum_{d_n} T_k(d_n | d_{\pi_k(n)}) = 1$. This table is easily computed using probability propagation in the two trees to compute the two marginals needed in the difference.

The optimal conditional probability table for a variable is *independent* of the parent-child relationships in the remainder of the network. So, for the current symptom, we compute the optimal conditional probability tables for all $N(N-1)/2$ possible parent-child relationships in $\mathcal{O}(N^2)$ time using probability propagation. Then, we use a minimum-weight directed spanning tree algorithm (Bock 1971) to search for the best tree.

Once all of the symptoms have been absorbed, we use the final tree distribution, $T_J(\mathbf{d})$ to make inferences about $\mathbf{d}$ given $\mathbf{s}$. The order in which the symptoms are absorbed will generally affect the quality of the resulting tree (Jaakkola and Jordan 1999), but we used a random ordering in the experiments reported below.

## 4 Results on QMR-DT type networks

Using the structural and parameter statistics of the QMR-DT network given in Shwe *et al.* (1991) we simulated 30 QMR-DT type networks with roughly 600 diseases each. There were 10 networks in each of 3 groups with 5, 10 and 15 instantiated active symptoms. We chose the number of active symptoms to be small enough that we can compare our approximate method with the exact QuickScore method (Heckerman 1989). We also tried two other approximate inference methods: local probability propagation (Murphy *et al.* 1999) and a variational upper bound (Jaakkola and Jordan 1999).

For medical diagnosis, an important question is how many most probable diseases $n'$ under the approximate posterior must be examined before the most probable $n$ diseases under the exact posterior are found. Clearly, $n \le n' \le N$. An exact inference algorithm will give $n' = n$, whereas an approximate algorithm that mistakenly ranks the most probable disease last will give $n' = N$. For each group of networks and each inference method, we averaged the 10 values of $n'$ for each value of $n$.

The left column of plots in Fig. 3 shows the average of $n'$ versus $n$ for 5, 10 and 15 active symptoms. The sequential tree-fitting method is closest to optimal ($n' = n$) in all cases. The right column of plots shows the "extra work" caused by the excess number of diseases $n' - n$ that must be examined for the approximate methods.

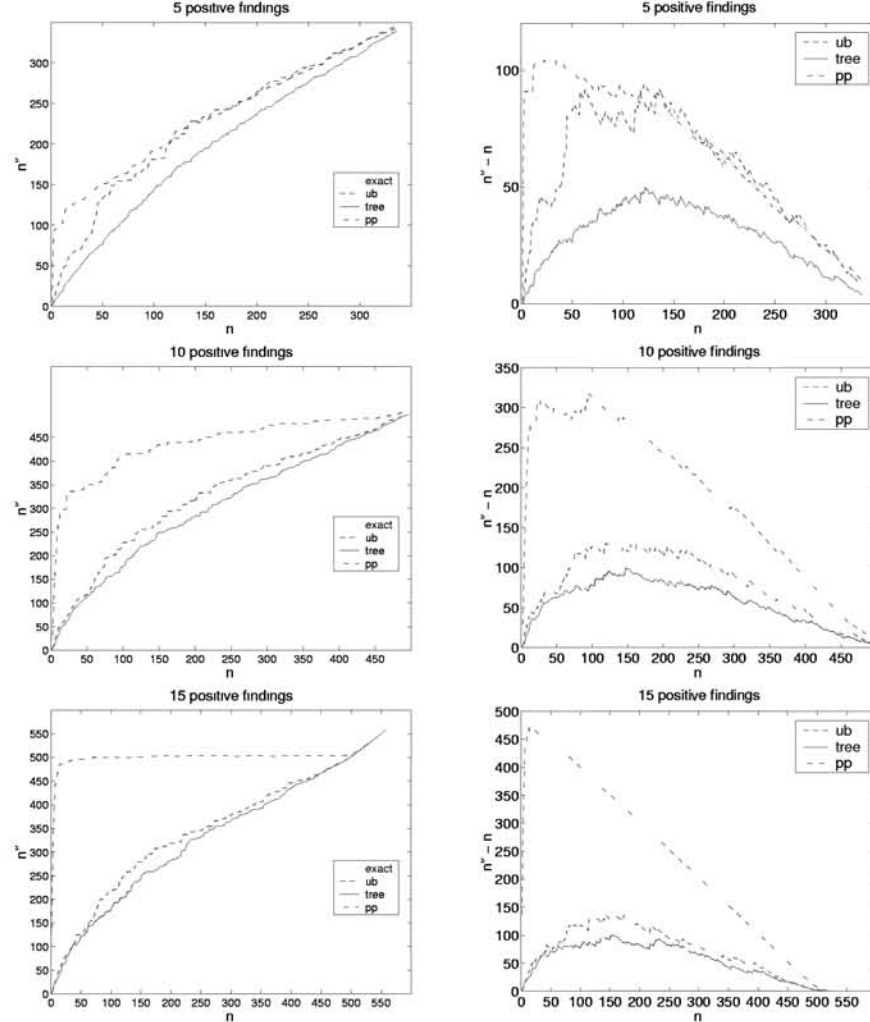

Figure 3: Comparisons of the number of most probable diseases $n'$ under the approximate posterior that must be examined before the most probable $n$ diseases under the exact posterior are found. Approximate methods include the sequential tree-fitting method presented in this paper (tree), local probability propagation (pp) and a variational upper bound (ub).

## 5  Summary

Noisy-OR networks can be used to model a variety of problems, including medical diagnosis. Exact inference in large, richly connected noisy-OR networks is intractable, and most approximate inference algorithms tend to concentrate on a small number of most probable configurations of the hidden variables under the posterior. We presented an "inclusive" variational method for bipartite noisy-OR networks that favors including all probable configurations, at the cost of including some improbable configurations. The method fits a tree to the posterior distribution sequentially, *i.e.*, one observation at a time. Results on an ensemble of QMR-DT type networks show that the method performs better than local probability propagation and a variational upper bound for ranking most probable diseases.

**Acknowledgements.** We thank Dale Schuurmans for discussions about this work.

**References**

H. Attias 1999. Independent factor analysis. *Neural Computation* **11:4**, 803–852.

F. Bock 1971. An algorithm to construct a minimum directed spanning tree in a directed network. *Developments in Operations Research*, Gordon and Breach, New York, 29–44.

W. T. Freeman and Y. Weiss 2001. On the fixed points of the max-product algorithm. To appear in *IEEE Transactions on Information Theory, Special issue on Codes on Graphs and Iterative Algorithms*.

B. J. Frey 1998. *Graphical Models for Machine Learning and Digital Communication*. MIT Press, Cambridge, MA.

B. J. Frey 2000. Filling in scenes by propagating probabilities through layers and into appearance models. *Proceedings of the IEEE Conference on Computer Vision and Pattern Recognition*, IEEE Computer Society Press, Los Alamitos, CA.

B. J. Frey and G. E. Hinton 1999. Variational learning in non-linear Gaussian belief networks. *Neural Computation* **11:1**, 193–214.

R. G. Gallager 1963. *Low-Density Parity-Check Codes*. MIT Press, Cambridge, MA.

Z. Ghahramani and M. I. Jordan 1997. Factorial hidden Markov models. *Machine Learning* **29**, 245–273.

D. Heckerman 1989. A tractable inference algorithm for diagnosing multiple diseases. *Proceedings of the Fifth Conference on Uncertainty in Artificial Intelligence*.

T. S. Jaakkola and M. I. Jordan 1999. Variational probabilistic inference and the QMR-DT network. *Journal of Artificial Intelligence Research* **10**, 291–322.

M. I. Jordan, Z. Ghahramani, T. S. Jaakkola and L. K. Saul 1999. An introduction to variational methods for graphical models. In M. I. Jordan (ed) *Learning in Graphical Models*, MIT Press, Cambridge, MA.

D. J. C MacKay 1999a. Good error-correcting codes based on very sparse matrices. *IEEE Transactions on Information Theory* **45:2**, 399–431.

D. J. C MacKay 1999b. Introduction to Monte Carlo methods. In M. I. Jordan (ed) *Learning in Graphical Models*, MIT Press, Cambridge, MA.

R. J. McEliece, E. R. Rodemich and J.-F. Cheng 1996. The turbo decision algorithm. *Proceedings of the $33^{rd}$ Allerton Conference on Communication, Control and Computing*, Champaign-Urbana, IL.

K. P. Murphy, Y. Weiss and M. I. Jordan 1999. Loopy belief propagation for approximate inference: An empirical study. *Proceedings of the Fifteenth Conference on Uncertainty in Artificial Intelligence*, Morgan Kaufmann, San Francisco, CA.

R. M. Neal 1993. Probabilistic inference using Markov chain Monte Carlo methods. Technical Report CRG-TR-93-1, Computer Science, University of Toronto.

R. M. Neal 1996. Sampling from multimodal distributions using tempered transitions. *Statistics and Computing* **6**, 353–366.

L. K. Saul, T. Jaakkola and M. I. Jordan 1996. Mean field theory for sigmoid belief networks. *Journal of Artificial Intelligence Research* **4**, 61–76.

L. K. Saul and M. I. Jordan 1996. Exploiting tractable substructures in intractable networks. In D. Touretzky, M. Mozer, and M. Hasselmo (eds) *Advances in Neural Information Processing Systems 8*. MIT Press, Cambridge, MA.

M. Shwe, B. Middleton, D. Heckerman, M. Henrion, E. Horvitz, H. Lehmann and G. Cooper 1991. Probabilistic diagnosis using a reformulation of the INTERNIST-1/QMR knowledge base I. The probabilistic model and inference algorithms. *Methods of Information in Medicine* **30**, 241–255.

## Footnotes

[1]However, the diseases are $dependent$ given that some symptoms are present.
